# The Noisy Euclidean Traveling Salesman Problem and Learning

**Mikio L. Braun, Joachim M. Buhmann**
braunm@cs.uni-bonn.de, jb@cs.uni-bonn.de
Institute for Computer Science, Dept. III,
University of Bonn
Römerstraße 164, 53117 Bonn, Germany

## Abstract

We consider noisy Euclidean traveling salesman problems in the plane, which are random combinatorial problems with underlying structure. Gibbs sampling is used to compute average trajectories, which estimate the underlying structure common to all instances. This procedure requires identifying the exact relationship between permutations and tours. In a learning setting, the average trajectory is used as a model to construct solutions to new instances sampled from the same source. Experimental results show that the average trajectory can in fact estimate the underlying structure and that overfitting effects occur if the trajectory adapts too closely to a single instance.

## 1 Introduction

The approach in combinatorial optimization is traditionally single-instance and worst-case-oriented. An algorithm is tested against the worst possible single instance. In reality, algorithms are often applied to a large number of related instances, the average-case performance being the measurement of interest. This constitutes a completely different problem: given a set of similar instances, construct solutions which are good on average. We call this kind of problem multiple-instances and average-case-oriented. Since the instances share some information, it might be expected that this problem is simpler than solving all instances separately, even for NP-hard problems.

We will study the following example of a multiple-instance average-case problem, which is built from the Euclidean travelings salesman problem (TSP) in the plane. Consider a salesman who makes weekly trips. At the beginning of each week, the salesman has a new set of appointments for the week, for which he has to plan the shortest round-trip. The location of the appointments will not be completely random, because there are certain areas which have a higher probability of containing an appointment, for example cities or business districts within cities. Instead of solving the planning problem each week from scratch, a clever salesman will try to exploit the underlying density and have a rough trip pre-planned, which he will only adapt from week to week.

An idealizing formulization of this setting is as follows. Fix the number of appointments $n \in \mathbb{N}$. Let $x_1, \ldots, x_n \in \mathbb{R}^2$ and $\sigma \in \mathbb{R}_+$. Then, the locations of the

appointments for each week are given as samples from the normally distributed random vectors ($i \in \{1, \ldots, n\}$)

$$X_i \sim \mathcal{N}(x_i, \sigma^2). \tag{1}$$

The random vector $(X_1, \ldots, X_n)$ will be called a *scenario*, sampled appointment locations *(sampled) instance*. The task consists in finding the permutation $\pi \in \mathsf{S}_n$ which minimizes $\pi \mapsto d_{\pi(n)\pi(1)} + \sum_{i=1}^{n-1} d_{\pi(i)\pi(i+1)}$, where $d_{ij} := \|X_i - X_j\|_2$, and $\mathsf{S}_n$ being the set of all bijective functions on the set $\{1, \ldots, n\}$. Typical examples are depicted in figure 1(a)–(c).

It turns out that the multiple-instance average-case setting is related to learning theory, especially to the theory of cost-based unsupervised learning. This relationship becomes clear if one considers the performance measure of interest. The algorithm takes a set of instances $I_1, \ldots, I_n$ as input and outputs a number of solutions $s_1, \ldots, s_n$. It is then measured by the average performance $(1/n) \sum_{k=1}^{n} C(s_k, I_k)$, where $C(s, I)$ denotes the cost of solution $s$ on instance $I$. We now modify the performance measure as follows. Given a finite number of instances $I_1, \ldots, I_n$, the algorithm has to construct a solution $s'$ on a newly sampled instance $I'$. The performance is then measured by the expected cost $E(C(s', I'))$. This can be interpreted as a learning task. The instances $I_1, \ldots, I_n$ are then the training data, $E(C(s', I'))$ is the analogue of the expected risk or cost, and the set of solutions is identified with the hypothesis class in learning theory.

In this paper, the setting presented in the previous paragraph is studied with the further restriction that only one training instance is present. From this training instance, an average solution is constructed, represented by a closed curve in the plane. This average trajectory is supposed to capture the essential structure of the underlying probability density, similar to the centroids in $K$-means clustering. Then, the average trajectory is used as a seed for a simple heuristic which constructs solutions on newly drawn instances. The average trajectories are computed by geometrically averaging tours which are drawn by a Gibbs sampler at finite temperature. This will be discussed in detail in sections 2 and 3. It turns out that the temperature acts as a scale or smoothing parameter. A few comments concerning the selection of this parameter are given in section 6.

The technical content of our approach is reminiscent of the "elastic net"-approaches of Durbin and Willshaw (see [2], [5]), but differs in many points. It is based on a completely different algorithmic approach using Gibbs sampling and a general technique for averaging tours. Our algorithm has polynomial complexity per Monte Carlo step and convergence is guaranteed by the usual bounds for Markov Chain Monte Carlo simulation and Gibbs sampling. Furthermore, the goal is not to provide a heuristic for computing the best solution, but to extract the relevant statistics of the Gibbs distribution at finite temperatures to generate the average trajectory, which will be used to compute solutions on future instances.

## 2 The Metropolis algorithm

The Metropolis algorithm is a well-known algorithm which simulates a homogeneous Markov chain whose distribution converges to the Gibbs distribution. We assume that the reader is familiar with the concepts, we give here only a brief sketch of the relevant results and refer to [6], [3] for further details.

Let $M$ be a finite set and $f \colon M \to \mathbb{R}$. The Gibbs distribution at temperature $T \in \mathbb{R}_+$ is given by ($m \in M$)

$$g_T(m) := \frac{\exp(-f(m)/T)}{\sum_{m' \in M} \exp(-f(m')/T)}. \tag{2}$$

The Metropolis algorithm works as follows. We start with any element $m \in M$ and set $X_1 \leftarrow m$. For $i \geq 2$, apply a random local update $m' := \phi(X_i)$. Then set

$$X_{i+1} \leftarrow \begin{cases} m' & \text{with probability } \min\left\{1, \exp\left(-(f(X_i) - f(m'))/T\right)\right\} \\ X_i & \text{else} \end{cases}. \quad (3)$$

This scheme converges to the Gibbs distribution if certain conditions on $\phi$ are met. Furthermore, a $L^2$-law of large numbers holds: For $h \colon M \to \mathbb{R}$, $\frac{1}{n} \sum_{k=1}^n h(X_k) \to \sum_{m \in M} g_T(m)h(m)$ in $L^2$. For TSP, $M = \mathsf{S}_n$ and $\phi$ is the Lin-Kernighan two-change [4], which consists in choosing two indexes $i, j$ at random and reversing the path between the appointments $i$ and $j$. Note that the Lin-Kernighan two-change and its generalizations for neighborhood search are powerful heuristic in itself.

## 3 Averaging Tours

Our goal is to compute the average trajectory, which should grasp the underlying structure common to all instances, with respect to the Gibbs measure at non-zero temperature $T$. The Metropolis algorithm produces a sequence of permutations $\pi_1, \pi_2, \ldots$ with $P\{\pi_n = .\} \to g_T(.)$ for $n \to \infty$. Since permutations cannot be added, we cannot simply compute the empirical means of $\pi_n$. Instead, we map permutations to their corresponding trajectories.

**Definition 1** (trajectory) *The trajectory of $\pi \in \mathsf{S}_n$ given $n$ points $x_1, \ldots, x_n$ is a mapping $\Gamma(\pi) \colon \{1, \ldots, n\} \to \mathbb{R}^2$ defined by $\Gamma(\pi)(i) := x_{\pi(i)}$. The set of all trajectories (for all sets of $n$ points) is denoted by $\mathsf{T}_n$ (this is the set of all mappings $\gamma \colon \{1, \ldots, n\} \to \mathbb{R}^2$).*

Addition of trajectories and multiplication with scalars can be defined pointwise. Then it is technically possible to compute $\frac{1}{k} \sum_{k=1}^n \Gamma(\pi_k)$. Unfortunately, this does not yield the desired results, since the relation between permutations and tours is not one-to-one. For example, the permutation obtained by starting the tour at a different city still corresponds to the same tour. We therefore need to define the addition of trajectories in a way which is independent of the choice of permutation (and therefore trajectory) to represent the tour. We will study the relationship between tours and permutations first in some detail, since we feel that the concepts introduced here might be generally useful for analyzing combinatorial optimization problems.

**Definition 2** (tour and length of a tour) *Let $G = (V, E)$ be a complete (undirected) graph with $V = \{1, \ldots, n\}$ and $E = \binom{V}{2}$. A subset $t \in E$ is called a tour iff $|t| = n$, for every $v \in V$, there exist exactly two $e_1, e_2 \in t$ such that $v \in e_1$ and $v \in e_2$, and $(V, t)$ is connected. Given a symmetric matrix $(d_{ij})$ of distances, the length of a tour $t$ is defined by $\ell(t) := \sum_{\{i,j\} \in t} d_{ij}$.*

The tour corresponding to a permutation $\pi \in \mathsf{S}_n$ is given by

$$\mathsf{t}(\pi) := \left\{\{\pi(1), \pi(n)\}\right\} \cup \bigcup_{i=1}^{n-1} \left\{\{\pi(i), \pi(i+1)\}\right\}. \quad (4)$$

If $\mathsf{t}(\pi) = t$ for a permutation $\pi$ and a tour $t$, we say that $\pi$ *represents* $t$. We call two permutations $\pi, \pi'$ *equivalent*, if they represent the same tour and write $\pi \sim \pi'$. Let $[\pi]$ denote the equivalence class of $\pi$ as usual. Note that the length of a permutation is fully determined by its equivalence class. Therefore, $\sim$ describes the intrinsic symmetries of the TSP formulated as an optimization problem on $\mathsf{S}_n$, denoted by $\mathrm{TSP}(\mathsf{S}_n)$.

We have to define the addition $\oplus$ of trajectories such that the sum is independent of the representation. This means that for two tours $t_1, t_2$ such that $t_1$ is represented

by $\pi_1$, $\pi_1'$ and $t_2$ by $\pi_2$, $\pi_2'$ it holds that $\Gamma(\pi_1) \oplus \Gamma(\pi_2) \sim \Gamma(\pi_1') \oplus \Gamma(\pi_2')$. The idea will be to normalize both summands before addition. We will first study the exact representation symmetry of $\mathrm{TSP}(\mathsf{S}_n)$.

**The $\mathrm{TSP}(\mathsf{S}_n)$ symmetry group**   Algebraically speaking, $\mathsf{S}_n$ is a group with concatenation of functions as multiplication, so we can characterize the equivalence classes of $\sim$ by studying the set of operations on a permutation which map to the same equivalent class. We define a group action of $\mathsf{S}_n$ on itself by right translation $(\pi, g \in \mathsf{S}_n)$:

$$\text{`` } \cdot \text{ ''} : \mathsf{S}_n \times \mathsf{S}_n \to \mathsf{S}_n, \qquad g \cdot \pi := \pi g^{-1}. \tag{5}$$

Note that any permutation in $\mathsf{S}_n$ can be mapped to another by an appropriate group action (namely $\pi \to \pi'$ by $(\pi'^{-1}\pi) \cdot \pi$.), such that the group action of $\mathsf{S}_n$ on itself suffices to study the equivalence classes of $\sim$.

For certain $g \in \mathsf{S}_n$, it holds that $\mathsf{t}(g \cdot \pi) = \mathsf{t}(\pi)$. We want to determine the maximal set $H_{\mathsf{t}}$ of elements which keeps $\mathsf{t}$ invariant. It even holds that $H_{\mathsf{t}}$ is a subgroup of $\mathsf{S}_n$: The identity is trivially in $H_{\mathsf{t}}$. Let $g, h$ be $\mathsf{t}$-invariant, then $\mathsf{t}((gh^{-1}) \cdot \pi) = \mathsf{t}(g \cdot (h^{-1} \cdot \pi)) = \mathsf{t}(h^{-1} \cdot \pi) = \mathsf{t}(h \cdot (h^{-1} \cdot \pi)) = \mathsf{t}(\pi)$. $H_{\mathsf{t}}$ will be called the *symmetry group* of $\mathrm{TSP}(\mathsf{S}_n)$ and it follows that $[\pi] = H_{\mathsf{t}} \cdot \pi := \{h \cdot \pi \mid h \in H_{\mathsf{t}}\}$.

The *shift* $\sigma$ and *reversal* $\varrho$ are defined by $(i \in \{1, \ldots, n\})$

$$\sigma(i) := \begin{cases} i+1 & i < n, \\ 1 & i = n \end{cases}, \qquad \varrho(i) := n + 1 - i, \tag{6}$$

and set $H := \langle \varrho, \sigma \rangle$, the group generated by $\sigma$ and $\varrho$. It holds that (this result is an easy consequence of $\varrho\varrho = \mathrm{id}_{\{1,\ldots,n\}}$, $\varrho\sigma = \sigma^{-1}\varrho$ and $\sigma^n = \mathrm{id}_{\{1,\ldots,n\}}$)

$$H = \{\sigma^k \mid k \in \{1, \ldots, n\}\} \cup \{\varrho\sigma^k \mid k \in \{1, \ldots, n\}\}. \tag{7}$$

The fundamental result is

**Theorem 1** *Let* $\mathsf{t}$ *be the mapping which sends permutations to tours as defined in* (4). *Then,* $H_{\mathsf{t}} = H$, *where* $H_{\mathsf{t}}$ *is the set of all* $\mathsf{t}$-*invariant permutations and* $H$ *is defined in* (7).

**Proof:**   It is obvious that $H \subseteq H_{\mathsf{t}}$. Now, let $h^{-1} \in H_{\mathsf{t}}$. We are going to prove that $\mathsf{t}$-invariant permutations are completely defined by their values on 1 and 2. Let $h \in H_{\mathsf{t}}$ and $k := h(1)$. Then, $h(2) = \sigma(k)$ or $h(2) = \sigma^{-1}(k)$, because otherwise, $h$ would give rise to a link $\big\{\{\pi(h(1)), \pi(h(2))\}\big\} \notin \mathsf{t}(\pi)$. For the same reason, $h(3)$ must be mapped to $\sigma^{\pm 2}(k)$. Since $h$ must be bijective, $h(3) \neq h(1)$, so that the sign of the exponent must be the same as for $h(2)$. In general, $h(i) = \sigma^{\pm(i-1)}(k)$. Now note that for $i, k \in \{1, \ldots, n\}$, $\sigma^i(k) = \sigma^k(i)$ and therefore,

$$h = \begin{cases} \sigma^{k-1} & \text{if } h(i) = \sigma^{i-1}(k) \\ \varrho\sigma^{n-k} & \text{if } h(i) = \sigma^{-i+1}(k) \end{cases}. \qquad \square$$

**Adding trajectories**   We can now define equivalence for trajectories. First define a group action of $\mathsf{S}_n$ on $\mathsf{T}_n$ analogously to (5): the action of $h \in H_{\mathsf{t}}$ on $\gamma \in \mathsf{T}_n$ is given by $h \cdot \gamma := \gamma \circ h^{-1}$. Furthermore, we say that $\gamma \sim \eta$, if $H_{\mathsf{t}} \cdot \gamma = H_{\mathsf{t}} \cdot \eta$.

Our approach is motivated geometrically. We measure distances between trajectories as follows. Let $d \colon \mathbb{R}^2 \times \mathbb{R}^2 \to \mathbb{R}_+$ be a metric. Then define $(\gamma, \eta \in \mathsf{T}_n)$

$$d(\gamma, \eta) := \sum_{k=1}^{n} d(\gamma(k), \eta(k)). \tag{8}$$

Before adding two trajectories we will first choose equivalent representations $\gamma'$, $\eta'$ which minimize $d(\gamma', \eta')$. Because of the results presented so far, searching through

all equivalent trajectories is computationally tractable. Note that for $h \in H_t$, it holds that $d(h \cdot \gamma, h \cdot \eta) = d(\gamma, \eta)$ as $h$ only reorders the summands. It follows that it suffices to change the representations only for one argument, since $d(h \cdot \gamma, i \cdot \eta) = d(\gamma, h^{-1}i \cdot \eta)$. So the time complexity of one addition reduces to $2n$ computation of distances which involve $n$ subtractions each.

The *normalizing action* is defined by ($\gamma, \eta \in T_n$)

$$n_{\gamma\eta} := \operatorname*{argmin}_{n \in H_t} d(\gamma, n \cdot \eta). \tag{9}$$

Assuming that the normalizing action is unique[1], we can prove

**Theorem 2** *Let $\gamma$, $\eta$ be two trajectories, and $n_{\gamma\eta}$ the unique normalizing action as defined in (9). Then, the operation*

$$\gamma \oplus \eta := \gamma + n_{\gamma\eta} \cdot \eta \tag{10}$$

*is representation invariant.*

**Proof:** Let $\gamma' = g \cdot \gamma$, $\eta' = h \cdot \eta$ for $g, h \in H_t$. We claim that $n_{\gamma'\eta'} = g n_{\gamma\eta} h^{-1}$. The normalizing action is defined by

$$n_{\gamma'\eta'} = \operatorname*{argmin}_{n' \in H_t} d(\gamma', n' \cdot \eta') = \operatorname*{argmin}_{n' \in H_t} d(g \cdot \gamma, n'h \cdot \eta) = \operatorname*{argmin}_{n' \in H_t} d(\gamma, g^{-1}n'h \cdot \eta), \tag{11}$$

by inserting $g^{-1}$ parallelly before both arguments in the last step. Since the normalizing action is unique, it follows that for the $n'$ realizing the minimum it holds that $g^{-1}n'h = n_{\gamma\eta}$ and therefore $n' = n_{\gamma'\eta'} = g n_{\gamma\eta} h^{-1}$. Now, consider the sum

$$\gamma' \oplus \eta' = \gamma' + n_{\gamma'\eta'}\eta' = g \cdot \gamma + (g n_{\gamma\eta} h^{-1})h \cdot \eta = g \cdot (\gamma + n_{\gamma\eta}\eta) \sim \gamma \oplus \eta, \tag{12}$$

which proves the representation independence.  □

The sum of more than two trajectories can be defined by normalizing everything with respect to the first summand, so that empirical sums $\frac{1}{k} \bigoplus_{i=1}^{n} \Gamma(\pi_i)$ are now well-defined.

## 4    Inferring Solutions on New Instances

We transfer a trajectory to a new set of appointments $x_1, \ldots, x_n$ by computing the *relaxed tour* using the following finite-horizon adaption technique:

First of all, passing times $t_i$ for all appointments are computed. We extend the domain of a trajectory $\gamma$ from $\{1, \ldots, n\}$ to the interval $[1, n + 1)$ by linear interpolation. Then we define $t_i$ such that $\gamma(t_i)$ is the earliest point with minimal distance between appointment $x_i$ and the trajectory. The passing times can be calculated easily by simple geometric considerations. The permutation which sorts $(t_i)_{i=1}^{n}$ is the *relaxed solution* of $\gamma$ to $(x_i)$.

In a post-processing step, self-intersections are removed first. Then, segments of length $w$ are optimized by exhaustive search. Let $\pi$ be the relaxed solution. The path from $\pi(i)$ to $\pi(i + w + 2)$ (index addition is modulo $n$) is replaced by the best alternative through the appointments $\pi(i + 1), \ldots, \pi(i + w + 1)$. Iterate for all $i \in \{1, \ldots, n\}$ until there is no further improvement. Since this procedure has time complexity $w!n$, it can only be done efficiently for small $w$.

## 5 Experiments

For experiments, we used the following set-up: We took the $\|.\|_1$-norm to determine the normalizing action. Typical sample-sizes for the Markov chain Monte Carlo integration were 1000 with 100 steps in between to decouple consecutive samples. Scenarios were modeled after eq. (1), where the $x_i$ were chosen to form simple geometric shapes.

Average trajectories for different temperatures are plotted in figures 1(a)–(c). As the temperature decreases, the average trajectory converges to the trajectory of a single locally optimal tour. The graphs demonstrate that the temperature $T$ acts as a *smoothing* parameter.

To estimate the expected risk of an average trajectory, the post-processed relaxed (PPR) solutions were averaged over 100 new instances (see figure 1(d)–(g)) in order to estimate the expected costs. The costs of the best solutions are good approximations, within 5% of the average minimum as determined by careful simulated annealing. An interesting effect occurs: the expected costs have their minimum at non-zero temperature. The corresponding trajectories are plotted in figure 1(e),(f). They recover the structure of the scenario. In other words, average trajectories computed at temperatures which are too low, start to *overfit* to noise present only in the instance for which they were computed. So computation of the global optimum of a noisy combinatorial optimization problem might not be the right strategy, because the solutions might not reflect the underlying structure. Averaging over many suboptimal solutions provides much better statistics.

## 6 Selection of the Temperature

The question remains how to select the optimal temperature. This problem is essentially the same as determining the correct model complexity in learning theory, and therefore no fully satisfying answer is readily available. The problem is nevertheless suited for the application of the heuristic provided by the *empirical risk approximation* (ERA) framework [1], which will be briefly sketched here.

The main idea of ERA is to coarse-grain the set of hypotheses $\mathcal{M}$ by treating hypotheses as equivalent which are only slightly different. Hypotheses whose $\ell_1$ mutual distance (defined in a similar fashion as (8)) is smaller than the parameter $\gamma \in \mathbb{R}_+$ are considered statistically equivalent. Selecting a subset of solutions such that $\ell_1$-spheres of radius $\gamma$ cover $\mathcal{M}$ results in the coarse-grained hypothesis set $\mathcal{M}_\gamma$. VC-type large deviation bounds depending on the size of the coarse-grained hypothesis class can now be derived:

$$P\left\{C_2(m_\gamma) - \min_{m \in \mathcal{M}} C_2(m) > 2\varepsilon\right\} \leq 2|\mathcal{M}_\gamma| \sup_{m \in \mathcal{M}_\gamma} \exp\left(-\frac{n(\varepsilon - \gamma)^2}{\alpha_m + c(\varepsilon - \gamma)}\right), \quad (13)$$

$\alpha_m$ depending on the distribution. The bound weighs two competing effects. On the one hand, increasing $\gamma$ introduces a systematic bias in the estimation. On the other hand, decreasing $\gamma$ increases the cardinality of the hypothesis class. Given a confidence $\delta > 0$, the probability of being worse than $\varepsilon > 0$ on a second instance and $\gamma$ are linked. So an optimal coarsening $\gamma$ can be determined. ERA then advocates to either sample from the $\gamma$-sphere around the empirical minimizer or average over these solutions.

Now it is well known, that the Gibbs sampler is concentrated on solutions whose costs are below a certain threshold. Therefore, the ERA is suited for our approach. In the relating equation the log cardinality of the approximation set occurs, which is usually interpreted as microcanonical entropy. This relates back to statistical physics, the starting point of our whole approach. Now interpreting $\gamma$ as energy, we can compute the stop temperature from the optimal $\gamma$. Using the well-known

relation from statistical physics $\frac{d\,\mathrm{entropy}}{d\,\mathrm{energy}} = T^{-1}$, we can derive a lower bound on the optimal temperature depending on variance estimates of the specific scenario given.

## 7   Conclusion

In reality, optimization algorithms are often applied to many similar instances. We pointed out that this can be interpreted as a learning problem. The underlying structure of similar instances should be extracted and used in order reduce the computational complexity for computing solutions to related instances.

Starting with the noisy Euclidean TSP, the construction of average tours is studied in this paper, which involves determining the exact relationship between permutation and tours, and identifying the intrinsic symmetries of the TSP. We hope that this technique might prove to be useful for other applications in the field of averaging over solutions of combinatorial problems. The average trajectories are able to capture the underlying structure common to all instances. A heuristic for constructing solutions on new instances is proposed. An empirical study of these procedures is conducted with results satisfying our expectations.

In terms of learning theory, overfitting effects can be observed. This phenomenon points at a deep connection between combinatorial optimization problems with noise and learning theory, which might be bidirectional. On the one hand, we believe that *noisy* (in contrast to *random*) combinatorial optimization problems are dominant in reality. Robust algorithms could be built by first estimating the undistorted structure and then using this structure as a guideline for constructing solutions for single instances. On the other hand, hardness of efficient optimization might be linked to the inability to extract meaningful structure. These connections, which are subject of further studies, link statistical complexity to computational complexity.

## Acknowledgments

The authors would like to thank Naftali Tishby, Scott Kirkpatrick and Michael Clausen for their helpful comments and discussions.

## Footnotes

[1]Otherwise, perturb the locations of the appointments by infinitesimal changes.

## References

[1] J. M. Buhmann and M. Held. Model selection in clustering by uniform convergence bounds. *Advances in Neural Information Processing Systems*, 12:216–222, 1999.

[2] R. Durbin and D. Willshaw. An analogue approach to the travelling salesman problem using an elastic net method. *Nature*, 326:689–691, 1987.

[3] S. Kirkpatrick, C. D. Gelatt, and M. P. Vecchi. Optimisation by simulated annealing. *Science*, 220:671–680, 1983.

[4] S. Lin and B. Kernighan. An effective heuristic algorithm for the traveling salesman problem. *Operations Research*, 21:498–516, 1973.

[5] P.D. Simic. Statistical mechanics as the underlying theory of "elastic" and "neural" optimizations. *Network*, 1:89–103, 1990.

[6] G. Winkler. *Image Analysis, Random fields and Dynamic Monte Carlo Methods*, volume 27 of *Application of Mathematics*. Springer, Heidelberg, 1995.

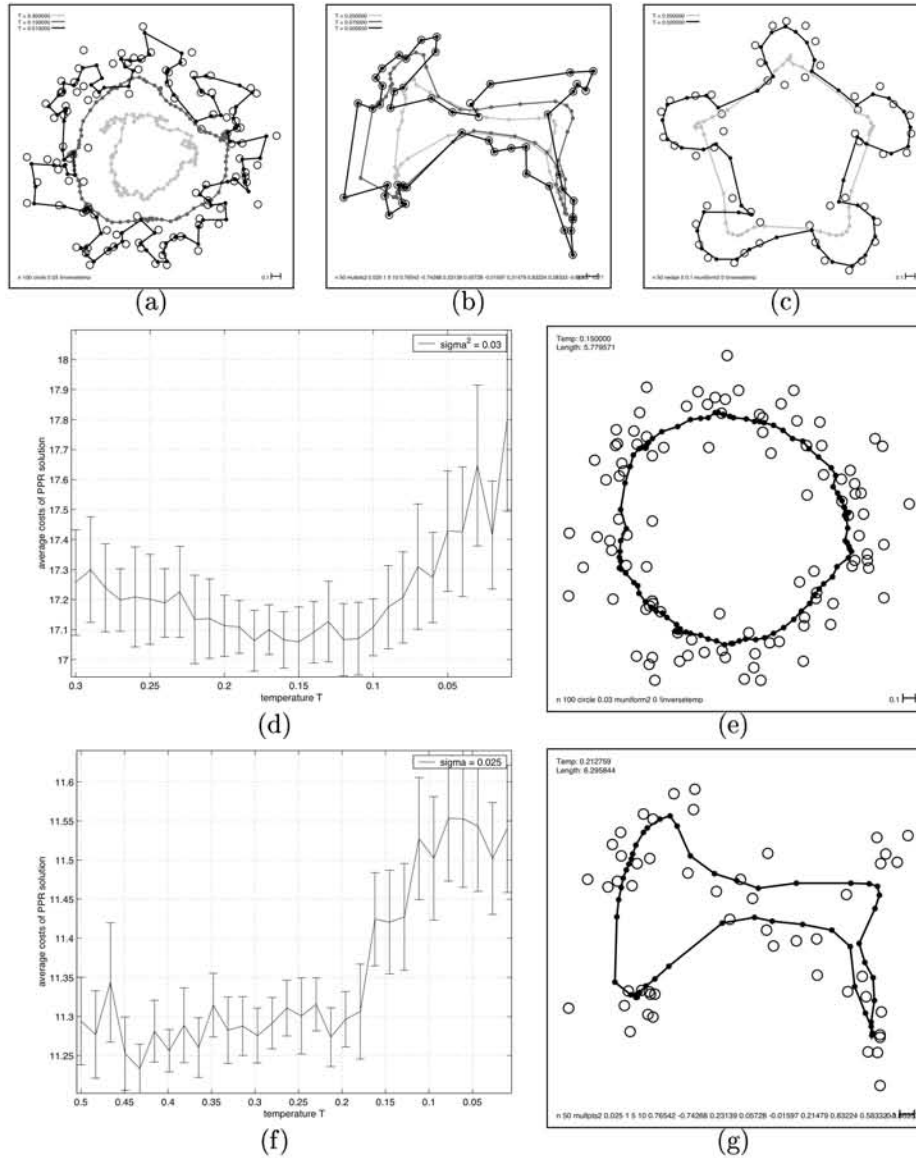

Figure 1: **(a)** Average trajectories at different temperatures for $n = 100$ appointments on a circle with $\sigma^2 = 0.03$. **(b)** Average trajectories at different temperatures, for multiple Gaussian sources, $n = 50$ and $\sigma^2 = 0.025$. **(c)** The same for an instance with structure on two levels. **(d)** Average tour length of the post-processed relaxed (PPR) solutions for the circle instance plotted in (a). The PPR width was $w = 5$. The average fits to noise in the data if the temperature is too low, leading to overfitting phenomena. Note that the average best solution is $\leq 16.5$. **(e)** The average trajectory with the smallest average length of its PPR solutions in (d). **(f)** Average tour length as in (d). The average best solution is $\leq 10.80$. **(g)** Lowest temperature trajectory with small average PPR solution length in (f).